# Phase transition in the family of $p$-resistances

**Morteza Alamgir**
Max Planck Institute for Intelligent Systems
Tübingen, Germany
morteza@tuebingen.mpg.de

**Ulrike von Luxburg**
Max Planck Institute for Intelligent Systems
Tübingen, Germany
ulrike.luxburg@tuebingen.mpg.de

## Abstract

We study the family of $p$-resistances on graphs for $p \geq 1$. This family generalizes the standard resistance distance. We prove that for any fixed graph, for $p = 1$ the $p$-resistance coincides with the shortest path distance, for $p = 2$ it coincides with the standard resistance distance, and for $p \to \infty$ it converges to the inverse of the minimal $s$-$t$-cut in the graph. Secondly, we consider the special case of random geometric graphs (such as $k$-nearest neighbor graphs) when the number $n$ of vertices in the graph tends to infinity. We prove that an interesting phase transition takes place. There exist two critical thresholds $p^*$ and $p^{**}$ such that if $p < p^*$, then the $p$-resistance depends on meaningful global properties of the graph, whereas if $p > p^{**}$, it only depends on trivial local quantities and does not convey any useful information. We can explicitly compute the critical values: $p^* = 1 + 1/(d - 1)$ and $p^{**} = 1 + 1/(d - 2)$ where $d$ is the dimension of the underlying space (we believe that the fact that there is a small gap between $p^*$ and $p^{**}$ is an artifact of our proofs). We also relate our findings to Laplacian regularization and suggest to use $q$-Laplacians as regularizers, where $q$ satisfies $1/p^* + 1/q = 1$.

## 1 Introduction

The graph Laplacian is a popular tool for unsupervised and semi-supervised learning problems on graphs. It is used in the context of spectral clustering, as a regularizer for semi-supervised learning, or to compute the resistance distance on graphs. However, it has been observed that under certain circumstances, standard Laplacian-based methods show undesired artifacts. In the semi-supervised learning setting Nadler et al. (2009) showed that as the number of unlabeled points increases, the solution obtained by Laplacian regularization degenerates to a non-informative function. von Luxburg et al. (2010) proved that as the number of points increases, the resistance distance converges to a meaningless limit function. Independently of these observations, a number of authors suggested to generalize Laplacian methods. The observation was that the "standard" Laplacian methods correspond to a vector space setting with $L_2$-norms, and that it might be beneficial to work in a more general $L_p$ setting for $p \neq 2$ instead. See Bühler and Hein (2009) for an application to clustering and Herbster and Lever (2009) for an application to label propagation. In this paper we take up several of these loose ends and connect them.

The main object under study in this paper is the family of $p$-resistances, which is a generalization of the standard resistance distance. Our first major result proves that the family of $p$-resistances is very rich and contains several special cases. The general picture is that the smaller $p$ is, the more the resistance is concentrated on "short paths". In particular, the case $p = 1$ corresponds to the shortest path distance in the graph, the case $p = 2$ to the standard resistance distance, and the case $p \to \infty$ to the inverse $s$-$t$-mincut.

Second, we study the behavior of $p$-resistances in the setting of random geometric graphs like lattice graphs, $\varepsilon$-graphs or $k$-nearest neighbor graphs. We prove that as the sample size $n$ increases, there

are two completely different regimes of behavior. Namely, there exist two critical thresholds $p^*$ and $p^{**}$ such that if $p < p^*$, the $p$-resistances convey useful information about the global topology of the data (such as its cluster properties), whereas for $p > p^{**}$ the resistance distances approximate a limit that does not convey any useful information. We can explicitly compute the value of the critical thresholds $p^* := 1 + 1/(d-1)$ and $p^{**} := 1 + 1/(d-2)$. This result even holds independently of the exact construction of the geometric graph.

Third, as we will see in Section 5, our results also shed light on the Laplacian regularization and semi-supervised learning setting. As there is a tight relationship between $p$-resistances and graph Laplacians, we can reformulate the artifacts described in Nadler et al. (2009) in terms of $p$-resistances. Taken together, our results suggest that standard Laplacian regularization should be replaced by $q$-Laplacian regularization (where $q$ is such that $1/p^* + 1/q = 1$).

## 2   Intuition and main results

Consider an undirected, weighted graph $G = (V, E)$ with $n$ vertices. As is standard in machine learning, the edge weights are supposed to indicate similarity of the adjacent points (not distances). Denote the weight of edge $e$ by $w_e \geq 0$ and the degree of vertex $u$ by $d_u$. The length of a path $\gamma$ in the weighted graph is defined as $\sum_{e \in \gamma} 1/w_e$. In the electrical network interpretation, a graph is considered as a network where each edge $e \in E$ has resistance $r_e = 1/w_e$. The *effective resistance* (or *resistance distance*) $R(s, t)$ between two vertices $s$ and $t$ in the network is defined as the overall resistance one obtains when connecting a unit volt battery to $s$ and $t$. It can be computed in many ways, but the one most useful for our paper is the following representation in terms of flows (cf. Section IX.1 of Bollobas, 1998):

$$R(s,t) = \min\left\{ \sum_{e \in E} r_e i_e^2 \mid i = (i_e)_{e \in E} \text{ unit flow from } s \text{ to } t \right\}. \tag{1}$$

In von Luxburg et al. (2010) it has been proved that in many random graph models, the resistance distance $R(s, t)$ between two vertices $s$ and $t$ converges to the trivial limit expression $1/d_s + 1/d_t$ as the size of the graph increases. We now want to present some intuition as to how this problem can be resolved in a natural way. For a subset $M \subset E$ of edges we define the contribution of $M$ to the resistance $R(s, t)$ as the part of the sum in (1) that runs over the edges in $M$. Let $i^*$ be a flow minimizing (1). To explain our intuition we separate this flow into two parts: $R(s, t) = R(s, t)^{local} + R(s, t)^{global}$. The part $R(s, t)^{local}$ stands for the contribution of $i^*$ that stems from the edges in small neighborhoods around $s$ and $t$, whereas $R(s, t)^{global}$ is the contribution of the remaining edges (exact definition given below). A useful distance function is supposed to encode the global geometry of the graph, for example its cluster properties. Hence, $R(s, t)^{global}$ should be the most important part in this decomposition. However, in case of the standard resistance distance the contribution of the global part becomes negligible as $n \to \infty$ (for many different models of graph construction). This effect happens because as the graph increases, there are so many different paths between $s$ and $t$ that once the flow has left the neighborhood of $s$, electricity can flow "without considerable resistance". The "bottleneck" for the flow is the part that comes from the edges in the local neighborhoods of $s$ and $t$, because here the flow has to concentrate on relatively few edges. So the dominating part is $R(s, t)^{local}$.

In order to define a useful distance function, we have to ensure that the global part has a significant contribution to the overall resistance. To this end, we have to avoid that the flow is distributed over "too many paths". In machine learning terms, we would like to achieve a flow that is "sparser" in the number of paths it uses. From this point of view, a natural attempt is to replace the 2-norm-optimization problem (1) by a $p$-norm optimization problem for some $p < 2$. Based on this intuition, our idea is to replace the squares in the flow problem (1) by a general exponent $p \geq 1$ and define the following new distance function on the graph.

**Definition 1 ($p$-resistance)**   *On any weighted graph $G$, for any $p \geq 1$ we define*

$$R_p(s,t) := \min\left\{ \sum_{e \in E} r_e |i_e|^p \mid i = (i_e)_{e \in E} \text{ unit flow from } s \text{ to } t \right\}. \tag{$*$}$$

As it turns out, our newly defined distance function $R_p$ is closely related but not completely identical to the $p$-resistance $R_p^H$ defined by Herbster and Lever (2009). A discussion of this issue can be found in Section 6.1.

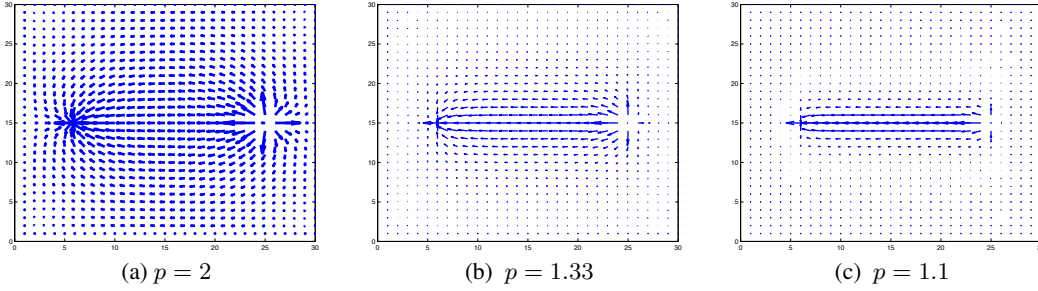

| (a) $p = 2$ | (b) $p = 1.33$ | (c) $p = 1.1$ |

Figure 1: The $s$-$t$-flows minimizing $(*)$ in a two-dimensional grid for different values of $p$. The smaller $p$, the more the flow concentrates along the shortest path.

In toy simulations we can observe that the desired effect of concentrating the flow on fewer paths takes place indeed. In Figure 1 we show how the optimal flow between two points $s$ and $t$ gets propagated through the network. We can see that the smaller $p$ is, the more the flow is concentrated along the shortest path between $s$ and $t$. We are now going to formally investigate the influence of the parameter $p$. Our first question is how the family $R_p(s, t)$ behaves as a function of $p$ (that is, on a fixed graph and for fixed $s, t$). The answer is given in the following theorem.

**Theorem 2 (Family of $p$-resistances)** *For any weighted graph $G$ the following statements are true:*

1. *For $p = 1$, the $p$-resistance coincides with the shortest path distance on the graph.*

2. *For $p = 2$, the $p$-resistance reduces to the standard resistance distance.*

3. *For $p \to \infty$, $R_p(s, t)^{p-1}$ converges to $1/m$ where $m$ is the unweighted $s$-$t$-mincut.*

This theorem shows that our intuition as outlined above was exactly the right one. The smaller $p$ is, the more flow is concentrated along straight paths. The extreme case is $p = 1$, which yields the shortest path distance. In the other direction, the larger $p$ is, the more widely distributed the flow is. Moreover, the theorem above suggests that for $p$ close to 1, $R_p$ encodes global information about the part of the graph that is concentrated around the shortest path. As $p$ increases, global information is still present, but now describes a larger portion of the graph, say, its cluster structure. This is the regime that is most interesting for machine learning. The larger $p$ becomes, the less global information is present in $R_p$ (because flows even use extremely long paths that take long detours), and in the extreme case $p \to \infty$ we are left with nothing but the information about the minimal $s$-$t$-cut. In many large graphs, the latter just contains local information about one of the points $s$ or $t$ (see the discussion at the end of this section). An illustration of the different behaviors can be found in Figure 2.

The next question, inspired by the results of von Luxburg et al. (2010), is what happens to $R_p(s, t)$ if we fix $p$ but consider a family $(G_n)_{n \in \mathbb{N}}$ of graphs such that the number $n$ of vertices in $G_n$ tends to $\infty$. Let us consider geometric graphs such as $k$-nearest neighbor graphs or $\varepsilon$-graphs. We now give exact definitions of the local and global contributions to the $p$-resistance. Let $r$ and $R$ be real numbers that depend on $n$ (they will be specified in Section 4) and $C \geq R/r$ a constant. We define the local neighborhood $\mathcal{N}(s)$ of vertex $s$ as the ball with radius $C \cdot r$ around $s$. We will see later that the condition $C \geq R/r$ ensures that $\mathcal{N}(s)$ contains at least all vertices adjacent to $s$. By abuse of notation we also write $e \in \mathcal{N}(s)$ if both endpoints of edge $e$ are contained in $\mathcal{N}(s)$. Let $i^*$ be the optimal flow in Problem $(*)$. We define

$$R_p^{local}(s) := \sum_{e \in \mathcal{N}(s)} r_e |i_e^*|^p,$$

$R_p^{local}(s, t) := R_p^{local}(s) + R_p^{local}(t)$, and $R_p^{global}(s, t) := R_p(s, t) - R_p^{local}(s, t)$. Our next result conveys that the behavior of the family of $p$-resistances shows an interesting phase transition. The statements involve a term $\tau_n$ that should be interpreted as the average degree in the graph $G_n$ (exact definition see later).

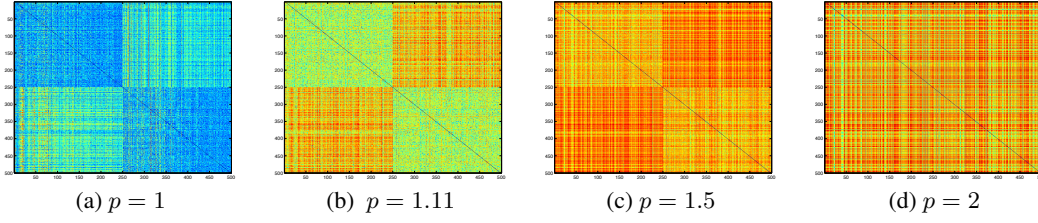

| (a) $p = 1$ | (b) $p = 1.11$ | (c) $p = 1.5$ | (d) $p = 2$ |

Figure 2: Heat plots of the $R_p$ distance matrices for a mixture of two Gaussians in $\mathbb{R}^{10}$. We can see that the larger $p$ it, the less pronounced the "global information" about the cluster structure is.

**Theorem 3 (Phase transition for $p$-resistances in large geometric graphs)** *Consider a family $(G_n)_{n \in \mathbb{N}}$ of unweighted geometric graphs on $\mathbb{R}^d$, $d > 2$ that satisfies some general assumptions (see Section 4 for definitions and details). Fix two vertices $s$ and $t$. Define the two critical values $p^* := 1 + 1/(d-1)$ and $p^{**} := 1 + 1/(d-2)$. Then, as $n \to \infty$, the following statements hold:*

*1. If $p < p^*$ and $\tau_n$ is sub-polynomial in $n$, then $R_p^{global}(s,t)/R_p^{local}(s,t) \to \infty$, that is the global contribution dominates the local one.*

*2. If $p > p^{**}$ and $\tau_n \to \infty$, then $R_p^{local}(s,t)/R_p^{global}(s,t) \to \infty$ and $R_p(s,t) \to \frac{1}{d_s^{p-1}} + \frac{1}{d_t^{p-1}}$, that is all global information vanishes.*

This result is interesting. It shows that there exists a non-trivial point of phase transition in the behavior of $p$-resistances: if $p < p^*$, then $p$-resistances are informative about the global topology of the graph, whereas if $p > p^{**}$ the $p$-resistances converge to trivial distance functions that do not depend on any global properties of the graph. In fact, we believe that $p^{**}$ should be $1 - 1/(d-1)$ as well, but our current proof leaves the tiny gap between $p^* = 1 - 1/(d-1)$ and $p^{**} = 1 - 1/(d-2)$.

Theorem 3 is a substantial extension of the work of von Luxburg et al. (2010), in several respects. First, and most importantly, it shows the complete picture of the full range of $p \geq 1$, and not just the single snapshot at $p = 2$. We can see that there is a range of values for $p$ for which $p$-resistance distances convey very important information about the global topology of the graph, even in extremely large graphs. Also note how nicely Theorems 2 and 3 fit together. It is well-known that as $n \to \infty$, the shortest path distance corresponding to $p = 1$ converges to the (geodesic) distance of $s$ and $t$ in the underlying space (Tenenbaum et al., 2000), which of course conveys global information. von Luxburg et al. (2010) proved that the standard resistance distance ($p = 2$) converges to the trivial local limit. Theorem 3 now identifies the point of phase transition $p^*$ between the boundary cases $p = 1$ and $p = 2$. Finally, for $p \to \infty$, we know by Theorem 2 that the $p$-resistance converges to the inverse of the $s$-$t$-min-cut. It is widely believed that the minimal $s$-$t$ cut in geometric graphs converges to the minimum of the degrees of $s$ and $t$ as $n \to \infty$ (even though a formal proof has yet to be presented and we cannot point to any reference). This is in alignment with the result of Theorem 3 that the $p$-resistance converges to $1/d_s^{p-1} + 1/d_t^{p-1}$. As $p \to \infty$, only the smaller of the two degrees contributes to the local part, which agrees with the limit for the $s$-$t$-mincut.

## 3 Equivalent optimization problems and proof of Theorem 2

In this section we will consider different optimization problems that are inherently related to $p$-resistances. All graphs in this section are considered to be weighted.

### 3.1 Equivalent optimization problems

Consider the following two optimization problems for $p > 1$:

**Flow-problem:** $\qquad R_p(s,t) := \min \left\{ \sum_{e \in E} r_e |i_e|^p \ \big| \ i = (i_e)_{e \in E} \text{ unit flow from } s \text{ to } t \right\}$ (∗)

**Potential problem:** $\quad C_p(s,t) = \min\left\{ \displaystyle\sum_{e=(u,v)} \frac{|\varphi(u) - \varphi(v)|^{1 + \frac{1}{p-1}}}{r_e^{\frac{1}{p-1}}} \,\Big|\, \varphi(s) - \varphi(t) = 1 \right\}$ $\quad (**)$

It is well known that these two problems are equivalent for $p = 2$ (see Section 1.3 of Doyle and Snell, 2000). We will now extend this result to general $p > 1$.

**Proposition 4 (Equivalent optimization problems)** *For $p > 1$, the following statements are true:*

1. *The flow problem $(*)$ has a unique solution.*

2. *The solutions of $(*)$ and $(**)$ satisfy $R_p(s,t) = (C_p(s,t))^{-\frac{1}{p-1}}$.*

To prove this proposition, we derive the Lagrange dual of problem $(*)$ and use the homogeneity of the variables to convert it to the form of problem $(**)$. Details can be found in the supplementary material. With this proposition we can now easily see why Theorem 2 is true.

**Proof of Theorem 2**. *Part (1)*. If we set $p = 1$, Problem $(*)$ coincides with the well-known linear programming formulation of the shortest path problem, see Chapter 12 of Bazaraa et al. (2010).
*Part (2)*. For $p = 2$, we get the well-known formula for the effective resistance.
*Part (3)*. For $p \to \infty$, the objective function in the dual problem $(**)$ converges to

$$C_\infty(s,t) := \min\left\{ \textstyle\sum_{e=(u,v)} |\varphi(u) - \varphi(v)| \,\big|\, \varphi(s) - \varphi(t) = 1 \right\}.$$

This coincides with the well-known linear programming formulation of the min-cut problem in unweighted graphs. Using Proposition 4 we finally obtain

$$\lim_{p\to\infty} R_p(s,t)^{p-1} = \lim_{p\to\infty} \frac{1}{C_p(s,t)} = \frac{1}{C_\infty(s,t)} = \frac{1}{\text{s-t-mincut}}.$$

## 4   Geometric graphs and the Proof of Theorem 3

In this section we consider the class of geometric graphs. The vertices of such graphs consist of points $X_1, .., X_n \in \mathbb{R}^d$, and vertices are connected by edges if the corresponding points are "close" (for example, they are $k$-nearest neighbors of each other). In most cases, we consider the set of points as drawn i.i.d from some density on $\mathbb{R}^d$. Consider the following general assumptions.

**General Assumptions:** *Consider a family $(G_n)_{n\in\mathbb{N}}$ of unweighted geometric graphs where $G_n$ is based on $X_1, ..., X_n \in M \subset \mathbb{R}^d$, $d > 2$. We assume that there exist $0 < r \le R$ (depending on $n$ and $d$) such that the following statements about $G_n$ holds simultaneously for all $x \in \{X_1, ..., X_n\}$:*

1. *Distribution of points: For $\rho \in \{r, R\}$ the number of sample points in $B(x, \rho)$ is of the order $\Theta(n \cdot \rho^d)$.*

2. *Graph connectivity: $x$ is connected to all sample points inside $B(x, r)$ and $x$ is not connected to any sample point outside $B(x, R)$.*

3. *Geometry of $M$: $M$ is a compact, connected set such that $M \setminus \partial M$ is still connected. The boundary $\partial M$ is regular in the sense that there exist positive constants $\alpha > 0$ and $\varepsilon_0 > 0$ such that if $\varepsilon < \varepsilon_0$, then for all points $x \in \partial M$ we have $\text{vol}(B_\varepsilon(x) \cap M) \ge \alpha\, \text{vol}(B_\varepsilon(x))$ (where $\text{vol}$ denotes the Lebesgue volume). Essentially this condition just excludes the situation where the boundary has arbitrarily thin spikes.*

It is a straightforward consequence of these assumptions that there exists some function $\tau(n) =: \tau_n$ such that $r$ and $R$ are both of the order $\Theta((\tau_n/n)^{1/d})$ and all degrees in the graph are of order $\Theta(\tau_n)$.

### 4.1   Lower and upper bounds and the proof of Theorem 3

To prove Theorem 3 we need to study the balance between $R_p^{local}$ and $R_p^{global}$. We introduce the shorthand notation

$$T_1 = \Theta\left(\frac{1}{n^{p(1-1/d)-1}\tau_n^{p(1+1/d)-1}}\right) \quad,\quad T_2 = \Theta\left(\frac{1}{\tau_n^{2(p-1)}} \sum_{k=1}^{1/r} \frac{1}{k^{(d-2)(p-1)}}\right).$$

**Theorem 5 (General bounds on $R_p^{local}$ and $R_p^{global}$)** *Consider a family $(G_n)_{n\in\mathbb{N}}$ of unweighted geometric graphs that satisfies the general assumptions. Then the following statements are true for any fixed pair $s, t$ of vertices in $G_n$:*

$$4C > R_p^{local}(s,t) \geq \frac{1}{d_s^{p-1}} + \frac{1}{d_t^{p-1}} \qquad and \qquad T_1 + T_2 \geq R_p^{global}(s,t) \geq T_1.$$

Note that by taking the sum of the two inequalities this theorem also leads to upper and lower bounds for $R_p(s,t)$ itself. The proof of Theorem 5 consists of several parts. To derive lower bounds on $R_p(s,t)$ we construct a second graph $G_n'$ which is a contracted version of $G_n$. Lower bounds can then be obtained by Rayleigh's monotonicity principle. To get upper bounds on $R_p(s,t)$ we exploit the fact that the $p$-resistance in an unweighted graph can be upper bounded by $\sum_{e\in E} i_e^p$, where $i$ is any unit flow from $s$ to $t$. We construct a particular flow that leads to a good upper bound. Finally, investigating the properties of lower and upper bounds we can derive the individual bounds on $R_p^{local}$ and $R_p^{global}$. Details can be found in the supplementary material.

Theorem 3 can now be derived from Theorem 5 by straight forward computations.

## 4.2 Applications

Our general results can directly be applied to many standard geometric graph models.

**The $\varepsilon$-graph**. We assume that $X_1, ..., X_n$ have been drawn i.i.d from some underlying density $f$ on $\mathbb{R}^d$, where $M := \mathrm{supp}(f)$ satisfies Part (3) of the general assumptions. Points are connected by unweighted edges in the graph if their Euclidean distances are smaller than $\varepsilon$. Exploiting standard results on $\varepsilon$-graphs (cf. the appendix in von Luxburg et al., 2010), it is easy to see that the general assumptions (1) and (2) are satisfied with probability at least $1 - c_1 n \exp(-c_2 n\varepsilon^d)$ (where $c_1, c_2$ are constants independent of $n$ and $d$) with $r = R = \varepsilon$ and $\tau_n = \Theta(n\varepsilon^d)$. The probability converges to 1 if $n \to \infty$, $\varepsilon \to 0$ and $n\varepsilon^d / \log(n) \to \infty$.

**$k$-nearest neighbor graphs.** We assume that $X_1, ..., X_n$ have been drawn i.i.d from some underlying density $f$ on $\mathbb{R}^d$, where $M := \mathrm{supp}(f)$ satisfies Part (3) of the general assumptions. We connect each point to its $k$ nearest neighbors by an undirected, unweighted edge. Exploiting standard results on kNN-graphs (cf. the appendix in von Luxburg et al., 2010), it is easy to see that the general assumptions (1) and (2) are satisfied with probability at least $1 - c_1 k \exp(-c_2 k)$ with $r = \Theta\big((k/n)^{1/d}\big)$, $R = \Theta\big((k/n)^{1/d}\big)$, and $\tau_n = k$. The probability converges to 1 if $n \to \infty$, $k \to \infty$, and $k/\log(n) \to \infty$.

**Lattice graphs.** Consider uniform lattices such as the square lattice or triangular lattice in $\mathbb{R}^d$. These lattices have constant degrees, which means that $\tau_n = \Theta(1)$. If we denote the edge length of grid by $\varepsilon$, the total number of nodes in the support will be in the order of $n = \Theta(1/\varepsilon^d)$. This means that the general assumptions hold for $r = R = \varepsilon = \Theta\big(\frac{1}{n^{1/d}}\big)$ and $\tau_n = \Theta(1)$. Note that while the lower bounds of Theorem 3 can be applied to the lattice case, our current upper bounds do not hold because they require that $\tau_n \to \infty$.

## 5 Regularization by $p$-Laplacians

One of the most popular methods for semi-supervised learning on graphs is based on Laplacian regularization. In Zhu et al. (2003) the label assignment problem is formulated as

$$\varphi = \mathrm{argmin}_x C(x) \qquad \text{subject to} \qquad \varphi(x_i) = y_i , \ i = 1, \dots, l \qquad (2)$$

where $y_i \in \{\pm 1\}$, $C(x) := \varphi^T L \varphi$ is the energy function involving the standard ($p = 2$) graph Laplacian $L$. This formulation is appealing and works well for small sample problems. However, Nadler et al. (2009) showed that the method is not well posed when the number of unlabeled data points is very large. In this setting, the solution of the optimization problem converges to a constant function with "spikes" at the labeled points. We now present a simple theorem that connects these findings to those concerning the resistance distance.

**Theorem 6 (Laplacian regularization in terms of resistance distance)** *Consider a semi-supervised classification problem with one labeled point per class: $\varphi(s) = 1$, $\varphi(t) = -1$. Denote*

*the solution of* (2) *by* $\varphi^*$, *and let* $v$ *be an unlabeled data point. Then*

$$\varphi^*(v) - \varphi^*(t) > \varphi^*(s) - \varphi^*(v) \iff R_2(v,t) > R_2(v,s).$$

*Proof.* It is easy to verify that $\varphi^* = L^\dagger(e_s - e_t)$ and $R_2(s,t) = (e_s - e_t)^T L^\dagger(e_s - e_t)$ where $L^\dagger$ is the pseudo-inverse of the Laplacian matrix $L$. Therefore we have $\varphi^*(v) = (e_v)^T L^\dagger(e_s - e_t)$ and

$$\varphi^*(v) - \varphi^*(t) > \varphi^*(s) - \varphi^*(v) \iff (e_v - e_t)^T L^\dagger(e_s - e_t) > (e_s - e_v)^T L^\dagger(e_s - e_t)$$

$$\stackrel{(a)}{\iff} (e_v - e_t)^T L^\dagger(e_v - e_t) > (e_v - e_s)^T L^\dagger(e_v - e_s) \iff R_2(v,t) > R_2(v,s).$$

Here in step (a) we use the symmetry of $L^\dagger$ to state that $e_v^T L^\dagger e_s = e_s^T L^\dagger e_v$.                     □

What does this theorem mean? We have seen above that in case $p = 2$, if $n \to \infty$,

$$R_2(v,t) \approx \frac{1}{d_v} + \frac{1}{d_t} \qquad \text{and} \qquad R_2(v,s) \approx \frac{1}{d_v} + \frac{1}{d_s}.$$

Hence, the theorem states that if we threshold the function $\varphi^*$ at 0 to separate the two classes, then all the points will be assigned to the labeled vertex with larger degree.

Our conjecture is that an analogue to Theorem 6 also holds for general $p$. For a precise formulation, define the matrix $\nabla$ as

$$\nabla_{i,j} = \begin{cases} \varphi(i) - \varphi(j) & i \sim j \\ 0 & \text{otherwise} \end{cases}$$

and introduce the matrix norm $\|A\|_{m,n} = \big( \sum_i ((\sum_j a_{ij}^m)^{1/m})^n \big)^{1/n}$. Consider $q$ such that $1/p + 1/q = 1$. We conjecture that if we used $\|\nabla\|_{q,q}$ as a regularizer for semi-supervised learning, then the corresponding solution $\varphi^*$ would satisfy

$$\varphi^*(v) - \varphi^*(t) > \varphi^*(s) - \varphi^*(v) \iff R_p(v,t) > R_p(v,s).$$

That is, the solution of the $q$-regularized problem would assign labels according to the $R_p$-distances. In particular, using $q$-regularization for the value $q$ with $1/q + 1/p^* = 1$ would resolve the artifacts of Laplacian regularization described in Nadler et al. (2009).

It is worth mentioning that this regularization is different from others in the literature. The usual Laplacian regularization term as in Zhu et al. (2003) coincides with $\|\nabla\|_{2,2}$, Zhou and Schölkopf (2005) use the $\|\nabla\|_{2,p}$ norm, and our conjecture is that the $\|\nabla\|_{q,q}$ norm would be a good candidate. Proving whether this conjecture is right or wrong is a subject of future work.

# 6 Related families of distance functions on graphs

In this section we sketch some relations between $p$-resistances and other families of distances.

## 6.1 Comparing Herbster's and our definition of $p$-resistances

For $p \leq 2$, Herbster and Lever (2009) introduced the following definition of $p$-resistances:

$$R_{p'}^H(s,t) := \frac{1}{C_{p'}^H(s,t)} \quad \text{with} \quad C_{p'}^H(s,t) := \min \Big\{ \sum_{e=(u,v)} \frac{|\varphi(u) - \varphi(v)|^{p'}}{r_e} \ \Big| \ \varphi(s) - \varphi(t) = 1 \Big\}.$$

In Section 3.1 we have seen that the potential and flow optimization problems are duals of each other. Based on this derivation we believe that the natural way of relating $R^H$ and $C^H$ would be to replace the $p'$ in Herbster's potential formulation by $q'$ such that $1/p' + 1/q' = 1$. That is, one would have to consider $C_{q'}^H$ and then define $\widehat{R}_{p'}^H := 1/C_{q'}^H$. In particular, reducing Herbster's $p'$ towards 1 has the same influence as increasing our $p$ to infinity and makes $R_{p'}^H$ converge to the minimal $s$-$t$-cut.

To ease further comparison, let us assume for now that we use "our" $p$ in the definition of Herbster's resistances. Then one can see by similar arguments as in Section 3.1 that $R_p^H$ can be rewritten as

$$R_p^H(s,t) := \min \Big\{ \sum_{e \in E} r_e^{p-1} |i_e|^p \ \Big| \ i = (i_e)_{e \in E} \text{ unit flow from } s \text{ to } t \Big\}. \tag{H}$$

Now it is easy to see that the main difference between Herbster's definition (H) and our definition (∗) is that (H) takes the power $p-1$ of the resistances $r_e$, while we keep the resistances with power 1. In many respects, $R_p$ and $R_p^H$ have properties that are similar to each other: they satisfy slightly different versions (with different powers or weights) of the triangle inequality, Rayleigh's monotonicity principle, laws for resistances in series and in parallel, and so on. We will not discuss further details due to space constraints.

## 6.2  Other families of distances

There also exist other families of distances on graphs that share some of the properties of $p$-resistances. We will only discuss the ones that are most related to our work, for more references see von Luxburg et al. (2010). The first such family was introduced by Yen et al. (2008), where the authors use a statistical physics approach to reduce the influence of long paths to the distance. This family is parameterized by a parameter $\theta$, contains the shortest path distance at one end ($\theta \to \infty$) and the standard resistance distance at the other end ($\theta \to 0$). However, the construction is somewhat ad hoc, the resulting distances cannot be computed in closed form and do not even satisfy the triangle inequality. A second family is the one of "logarithmic forest distances" by Chebotarev (2011). Even though its derivation is complicated, it has a closed form solution and can be interpreted intuitively: The contribution of a path to the overall distance is "discounted" by a factor $(1/\alpha)^l$ where $l$ is the length of the path. For $\alpha \to 0$, the logarithmic forest distance distance converges to the shortest path distance, for $\alpha \to \infty$, it converges to the resistance distance.

At the time of writing this paper, the major disadvantage of both the families introduced by Yen et al. (2008) and Chebotarev (2011) is that it is unknown how their distances behave as the size of the graph increases. It is clear that on the one end (shortest path), they convey global information, whereas on the other end (resistance distance) they depend on local quantities only when $n \to \infty$. But what happens to all intermediate parameter values? Do all of them lead to meaningless distances as $n \to \infty$, or is there some interesting phase transition as well? As long as this question has not been answered, one should be careful when using these distances. In particular, it is unclear how the parameters ($\theta$ and $\alpha$, respectively) should be chosen, and it is hard to get an intuition about this.

## 7  Conclusions

We proved that the family of $p$-resistances has a wide range of behaviors. In particular, for $p=1$ it coincides with the shortest path distance, for $p=2$ with the standard resistance distance and for $p \to \infty$ it is related to the minimal $s$-$t$-cut. Moreover, an interesting phase transition takes place: in large geometric graphs such as $k$-nearest neighbor graphs, the $p$-resistance is governed by meaningful global properties as long as $p < p^* := 1 + 1/(d-1)$, whereas it converges to the trivial local quantity $1/d_s^{p-1} + 1/d_t^{p-1}$ if $p > p^{**} := 1 + 1/(d-2)$. Our suggestion for practice is to use $p$-resistances with $p \approx p^*$. For this value of $p$, the $p$-resistances encode those global properties of the graph that are most important for machine learning, namely the cluster structure of the graph.

Our findings are interesting on their own, but also help in explaining several artifacts discussed in the literature. They go much beyond the work of von Luxburg et al. (2010) (which only studied the case $p=2$) and lead to an intuitive explanation of the artifacts of Laplacian regularization discovered in Nadler et al. (2009). An interesting line of future research will be to connect our results to the ones about $p$-eigenvectors of $p$-Laplacians (Bühler and Hein, 2009). For $p=2$, the resistance distance can be expressed in terms of the eigenvalues and eigenvectors of the Laplacian. We are curious to see whether a refined theory on $p$-eigenvalues can lead to similarly tight relationships for general values of $p$.

**Acknowledgements**

We would like to thank the anonymous reviewers who discovered an inconsistency in our earlier proof, and Bernhard Schölkopf for helpful discussions.

# References

M. Bazaraa, J. Jarvis, and H. Sherali. *Linear Programming and Network Flows*. Wiley-Interscience, 2010.

B. Bollobas. *Modern Graph Theory*. Springer, 1998.

T. Bühler and M. Hein. Spectral clustering based on the graph $p$-Laplacian. In *Proceedings of the International Conference on Machine Learning (ICML)*, pages 81–88, 2009.

P. Chebotarev. A class of graph-geodetic distances generalizing the shortets path and the resistance distances. *Discrete Applied Mathematics*, 159(295 – 302), 2011.

P. G. Doyle and J. Laurie Snell. Random walks and electric networks, 2000. URL `http://www.citebase.org/abstract?id=oai:arXiv.org:math/0001057`.

M. Herbster and G. Lever. Predicting the labelling of a graph via minimum p-seminorm interpolation. In *Conference on Learning Theory (COLT)*, 2009.

B. Nadler, N. Srebro, and X. Zhou. Semi-supervised learning with the graph Laplacian: The limit of infinite unlabelled data. In *Advances in Neural Information Processing Systems (NIPS)*, 2009.

J. Tenenbaum, V. de Silva, and J. Langford. Supplementary material to "A Global Geometric Framework for Nonlinear Dimensionality Reduction". *Science*, 290:2319 – 2323, 2000. URL `http://isomap.stanford.edu/BdSLT.pdf`.

U. von Luxburg, A. Radl, and M. Hein. Getting lost in space: Large sample analysis of the commute distance. In *Neural Information Processing Systems (NIPS)*, 2010.

L. Yen, M. Saerens, A. Mantrach, and M. Shimbo. A family of dissimilarity measures between nodes generalizing both the shortest-path and the commute-time distances. In *Proceedings of the 14th ACM SIGKDD International Conference on Knowledge Discovery and Data Mining*, pages 785–793, 2008.

D. Zhou and B. Schölkopf. Regularization on discrete spaces. In *DAGM-Symposium*, pages 361–368, 2005.

X. Zhu, Z. Ghahramani, and J. D. Lafferty. Semi-supervised learning using Gaussian fields and harmonic functions. In *ICML*, pages 912–919, 2003.

